# Online Submodular Minimization

**Elad Hazan**
IBM Almaden Research Center
650 Harry Rd, San Jose, CA 95120
hazan@us.ibm.com

**Satyen Kale**
Yahoo! Research
4301 Great America Parkway, Santa Clara, CA 95054
skale@yahoo-inc.com

## Abstract

We consider an online decision problem over a discrete space in which the loss function is submodular. We give algorithms which are computationally efficient and are Hannan-consistent in both the full information and bandit settings.

## 1  Introduction

Online decision-making is a learning problem in which one needs to choose a decision repeatedly from a given set of decisions, in an effort to minimize costs over the long run, even in the face of complete uncertainty about future outcomes. The performance of an online learning algorithm is measured in terms of its *regret*, which is the difference between the total cost of the decisions it chooses, and the cost of the optimal decision chosen in hindsight. A *Hannan-consistent* algorithm is one that achieves sublinear regret (as a function of the number of decision-making rounds). Hannan-consistency implies that the average per round cost of the algorithm converges to that of the optimal decision in hindsight.

In the past few decades, a variety of Hannan-consistent algorithms have been devised for a wide range of decision spaces and cost functions, including well-known settings such as prediction from expert advice [10], online convex optimization [15], etc. Most of these algorithms are based on an online version of convex optimization algorithms. Despite this success, many online decision-making problems still remain open, especially when the decision space is discrete and large (say, exponential size in the problem parameters) and the cost functions are non-linear.

In this paper, we consider just such a scenario. Our decision space is now the set of all subsets of a ground set of $n$ elements, and the cost functions are assumed to be *submodular*. This property is widely seen as the discrete analogue of convexity, and has proven to be a ubiquitous property in various machine learning tasks (see [4] for references). A crucial component in these latter results are the celebrated polynomial time algorithms for submodular function minimization [7].

To motivate the online decision-making problem with submodular cost functions, here is an example from [11]. Consider a factory capable of producing any subset from a given set of $n$ products $E$. Let $f : 2^E \mapsto \mathbb{R}$ be the cost function for producing any such subset (here, $2^E$ stands for the set of all subsets of $E$). Economics tells us that this cost function should satisfy the law of diminishing returns: i.e., the additional cost of producing an additional item is lower the more we produce. Mathematically stated, for all sets $S, T \subseteq E$ such that $T \subseteq S$, and for all elements $i \in E$, we have

$$f(T \cup \{i\}) - f(T) \ \geq \ f(S \cup \{i\}) - f(S).$$

Such cost functions are called *submodular*, and frequently arise in real-world economic and other scenarios. Now, for every item $i$, let $p_i$ be the market price of the item, which is only determined in the future based on supply and demand. Thus, the profit from producing a subset $S$ of the items is $P(S) = \sum_{i \in S} p_i - f(S)$. Maximizing profit is equivalent to minimizing the function $-P$, which is easily seen to be submodular as well.

The online decision problem which arises is now to decide which set of products to produce, to maximize profits in the long run, without knowing in advance the cost function or the market prices. A

more difficult version of this problem, perhaps more realistic, is when the only information obtained is the actual profit of the chosen subset of items, and no information on the profit possible for other subsets.

In general, the Online Submodular Minimization problem is the following. In each iteration, we choose a subset of a ground set of $n$ elements, and then observe a submodular cost function which gives the cost of the subset we chose. The goal is to minimize the regret, which is the difference between the total cost of the subsets we chose, and the cost of the best subset in hindsight. Depending on the feedback obtained, we distinguish between two settings, full-information and bandit. In the full-information setting, we can query each cost function at as many points as we like. In the bandit setting, we only get to observe the cost of the subset we chose, and no other information is revealed.

Obviously, if we ignore the special structure of these problems, standard algorithms for learning with expert advice and/or with bandit feedback can be applied to this setting. However, the computational complexity of these algorithms would be proportional to the number of subsets, which is $2^n$. In addition, for the submodular bandits problem, even the regret bounds have an exponential dependence on $n$. It is hence of interest to design *efficient* algorithms for these problems. For the bandit version an even more basic question arises: does there exist an algorithm with regret which depends only polynomially on $n$?

In this paper, we answer these questions in the affirmative. We give efficient algorithms for both problems, with regret which is bounded by a polynomial in $n$ – the underlying dimension – and sublinearly in the number of iterations. For the full information setting, we give two different randomized algorithms with expected regret $O(n\sqrt{T})$. One of these algorithms is based on the follow-the-perturbed-leader approach [5, 9]. We give a new way of analyzing such an algorithm. This analysis technique should have applications for other problems with large decision spaces as well. This algorithm is combinatorial, strongly polynomial, and can be easily generalized to arbitrary distributive lattices, rather than just all subsets of a given set. The second algorithm is based on convex analysis. We make crucial use of a continuous extension of a submodular function known as the *Lovász extension*. We obtain our regret bounds by running a (sub)gradient descent algorithm in the style of Zinkevich [15].

For the bandit setting, we give a randomized algorithm with expected regret $O(nT^{2/3})$. This algorithm also makes use of the Lovász extension and gradient descent. The algorithm folds exploration and exploitation steps into a single sample and obtains the stated regret bound. We also show that these regret bounds hold with high probability. Note that the technique of Flaxman, Kalai and McMahan [1], when applied to the Lovász extension, gives a worse regret bound of $O(nT^{3/4})$.

## 2 Preliminaries and Problem Statement

**Submodular functions.** The decision space is the set of all subsets of a universe of $n$ elements, $[n] = \{1, 2, \ldots, n\}$. The set of all subsets of $[n]$ is denoted $2^{[n]}$. For a set $S \subseteq [n]$, denote by $\chi_S$ its characteristic vector in $\{0,1\}^n$, i.e. $\chi_S(i) = 1$ if $i \in S$, and 0 otherwise.

A function $f : 2^{[n]} \to \mathbb{R}$ is called *submodular* if for all sets $S, T \subseteq [n]$ such that $T \subseteq S$, and for all elements $i \in E$, we have

$$f(T + i) - f(T) \ \geq \ f(S + i) - f(S).$$

Here, we use the shorthand notation $S + i$ to indicate $S \cup \{i\}$. An explicit description of $f$ would take exponential space. We assume therefore that the only way to access $f$ is via a *value oracle*, i.e. an oracle that returns the value of $f$ at any given set $S \subseteq [n]$.

Given access to a value oracle for a submodular function, it is possible to minimize it in polynomial time [3], and indeed, even in strongly polynomial time [3, 7, 13, 6, 12, 8]. The current fastest strongly polynomial algorithm are those of Orlin[12] and Iwata-Orlin [8], which takes time $O(n^5\text{EO} + n^6)$, where EO is the time taken to run the value oracle. The fastest weakly polynomial algorithm is that of Iwata [6] and Iwata-Orlin [8] which runs in time $\tilde{O}(n^4\text{EO} + n^5)$.

**Online Submodular Minimization.** In the Online Submodular Minimization problem, over a sequence of iterations $t = 1, 2, \ldots$, an online decision maker has to repeatedly chose a subset

$S_t \subseteq [n]$. In each iteration, after choosing the set $S_t$, the cost of the decision is specified by a submodular function $f_t : 2^{[n]} \to [-1, 1]$. The decision maker incurs cost $f_t(S_t)$. The *regret* of the decision maker is defined to be

$$\text{Regret}_T := \sum_{t=1}^{T} f_t(S_t) - \min_{S \subseteq [n]} \sum_{t=1}^{T} f_t(S).$$

If the sets $S_t$ are chosen by a randomized algorithm, then we consider the expected regret over the randomness in the algorithm.

An online algorithm to choose the sets $S_t$ will be said to be Hannan-consistent if it ensures that $\text{Regret}_T = o(T)$. The algorithm will be called *efficient* if it computes each decision $S_t$ in $\text{poly}(n, t)$ time. Depending on the kind of feedback the decision maker receives, we distinguish between two settings of the problem:

- **Full information setting.** In this case, in each round $t$, the decision maker has unlimited access to the value oracles of the previously seen cost function $f_1, f_2, \ldots f_{t-1}$.

- **Bandit setting.** In this case, in each round $t$, the decision maker only observes the cost of her decision $S_t$, viz. $f_t(S_t)$, and receives no other information.

**Main Results.** In the setup of the Online Submodular Minimization, we have the following results:

**Theorem 1.** *In the full information setting of Online Submodular Minimization, there is an efficient randomized algorithm that attains the following regret bound:*

$$\mathbf{E}[\text{Regret}_T] \leq O(n\sqrt{T}).$$

*Furthermore, $\text{Regret}_T \leq O((n + \sqrt{\log(1/\varepsilon)})\sqrt{T})$ with probability at least $1 - \varepsilon$.*

**Theorem 2.** *In the bandit setting of Online Submodular Minimization, there is an efficient randomized algorithm that attains the following regret bound:*

$$\mathbf{E}[\text{Regret}_T] \leq O(nT^{2/3}).$$

*Furthermore, $\text{Regret}_T \leq O(nT^{2/3}\sqrt{\log(1/\varepsilon)})$ with probability at least $1 - \varepsilon$.*

Both of the theorems above hold against both oblivious as well as adaptive adversaries.

**The Lovász Extension.** A major technical construction we need for the algorithms is the *Lovász extension* $\hat{f}$ of the submodular function $f$. This is defined on the Boolean hypercube $\mathcal{K} = [0, 1]^n$ and takes real values. Before defining the Lovász extension, we need the concept of a chain of subsets of $[n]$:

**Definition 1.** *A **chain** of subsets of $[n]$ is a collection of sets $A_0, A_1, \ldots, A_p$ such that*

$$A_0 \subset A_1 \subset A_2 \subset \cdots \subset A_p.$$

*A **maximal chain** is one where $p = n$. For a maximal chain, we have $A_0 = \emptyset$, $A_n = [n]$, and there is a unique associated permutation $\pi : [n] \to [n]$ such that for all $i \in [n]$, we have $A_{\pi(i)} = A_{\pi(i)-1} + i$.*

Now let $x \in \mathcal{K}$. There is a unique chain $A_0 \subset A_2 \subset \cdots A_p$ such that $x$ can be expressed as a convex combination $x = \sum_{i=0}^{p} \mu_i \chi_{A_i}$ where $\mu_i > 0$ and $\sum_{i=0}^{p} \mu_i = 1$. A nice way to construct this combination is the following random process: choose a threshold $\tau \in [0, 1]$ uniformly at random, and consider the level set $S_\tau = \{i : x_i > \tau\}$. The sets in the required chain are exactly the level sets which are obtained with positive probability, and for any such set $A_i$, $\mu_i = \mathbf{Pr}[S_\tau = A_i]$. In other words, we have $x = \mathbf{E}_\tau[\chi_{S_\tau}]$. This follows immediately by noting that for any $i$, we have $\mathbf{Pr}_\tau[i \in S_\tau] = x_i$. Of course, the chain and the weights $\mu_i$ can also be constructed deterministically simply by sorting the coordinates of $x$.

Now, we are ready to define[1] the Lovász extension $\hat{f}$:

**Definition 2.** *Let $x \in \mathcal{K}$. Let $A_0 \subset A_2 \subset \cdots A_p$ such that $x$ can be expressed as a convex combination $x = \sum_{i=0}^{p} \mu_i \chi_{A_i}$ where $\mu_i > 0$ and $\sum_{i=0}^{p} \mu_i = 1$. Then the value of the Lovász extension $\hat{f}$ at $x$ is defined to be*

$$\hat{f}(x) := \sum_{i=0}^{p} \mu_i f(A_i).$$

The preceding discussion gives an equivalent way of defining the Lovász extension: choose a threshold $\tau \in [0, 1]$ uniformly at random, and consider the level set $S_\tau = \{i : x_i > \tau\}$. Then we have

$$\hat{f}(x) = \mathbf{E}_\tau[f(S_\tau)].$$

Note that the definition immediately implies that for all sets $S \subseteq [n]$, we have $\hat{f}(\chi_S) = f(S)$.

We will also need the notion of a maximal chain associated to a point $x \in \mathcal{K}$ in order to define subgradients of the Lovász extension:

**Definition 3.** *Let $x \in \mathcal{K}$, and let $A_0 \subset A_2 \subset \cdots A_p$ be the unique chain such that $x = \sum_{i=0}^{p} \mu_i \chi_{A_i}$ where $\mu_i > 0$ and $\sum_{i=0}^{p} \mu_i = 1$. A **maximal chain associated with** $x$ is any maximal completion of the $A_i$ chain, i.e. a maximal chain $\emptyset = B_0 \subset B_1 \subset B_2 \subset \cdots B_n = [n]$ such that all sets $A_i$ appear in the $B_j$ chain.*

We have the following key properties of the Lovász extension. For proofs, refer to Fujishige [2], chapter IV.

**Proposition 3.** *The following properties of the Lovász extension $\hat{f} : \mathcal{K} \to \mathbb{R}$ hold:*

1. *$\hat{f}$ is convex.*

2. *Let $x \in \mathcal{K}$. Let $\emptyset = B_0 \subset B_1 \subset B_2 \subset \cdots B_n = [n]$ be an arbitrary maximal chain associated with $x$, and let $\pi : [n] \to [n]$ be the corresponding permutation. Then, a subgradient $g$ of $\hat{f}$ at $x$ is given as follows:*

$$g_i = f(B_{\pi(i)}) - f(B_{\pi(i)-1}).$$

## 3 The Full Information Setting

In this section we give two algorithms for regret minimization in the full information setting, both of which attain the same regret bound of $O(n\sqrt{T})$. The first is a randomized combinatorial algorithm, based on the "follow the leader" approach of Hannan [5] and Kalai-Vempala [9], and the second is an analytical algorithm based on (sub)gradient descent on the Lovász extension.

Both algorithms have pros and cons: while the second algorithm is much simpler and more efficient, we do not know how to extend it to distributive lattices, for which the first algorithm readily applies.

### 3.1 A Combinatorial Algorithm

In this section we analyze a combinatorial, strongly polynomial, algorithm for minimizing regret in the full information Online Submodular Minimization setting:

---
**Algorithm 1** Submodular Follow-The-Perturbed-Leader

---
1: Input: parameter $\eta > 0$.
2: Initialization: For every $i \in [n]$, choose a random number $r_i \in [-1/\eta, 1/\eta]$ uniformly at random. Define $R : 2^{[n]} \to \mathbb{R}$ as $R(S) = \sum_{i \in S} r_i$.
3: **for** $t = 1$ to $T$ **do**
4:     Use the set $S_t = \arg\min_{S \subseteq [n]} \sum_{\tau=1}^{t-1} f_\tau(S) + R(S)$, and obtain cost $f_t(S_t)$.
5: **end for**

---

Define $\Phi_t : 2^{[n]} \to \mathbb{R}$ as $\Phi_t(S) = \sum_{\tau=1}^{t-1} f_\tau(S) + R(S)$. Note that $R$ is a submodular function, and $\Phi_t$, being the sum of submodular functions, is itself submodular. Furthermore, it is easy to construct

a value oracle for $\Phi_t$ simply by using the value oracles for the $f_\tau$. Thus, the optimization in step 3 is poly-time solvable given oracle access to $\Phi_t$.

While the algorithm itself is a simple extension of Hannan's [5] follow-the-perturbed-leader algorithm, previous analysis (such as Kalai and Vempala [9]), which rely on linearity of the cost functions, cannot be made to work here. Instead, we introduce a new analysis technique: we divide the decision space using $n$ different cuts so that any two decisions are separated by at least one cut, and then we give an upper bound on the probability that the chosen decision switches sides over each such cut. This new technique may have applications to other problems as well. We now prove the regret bound of Theorem 1:

**Theorem 4.** *Algorithm 1 run with parameter $\eta = 1/\sqrt{T}$ achieves the following regret bound:*

$$\mathbf{E}[Regret_T] \leq 6n\sqrt{T}.$$

*Proof.* We note that the algorithm is essentially running a "follow-the-leader" algorithm on the cost functions $f_0, f_1, \ldots, f_{t-1}$, where $f_0 = R$ is a fictitious "period 0" cost function used for regularization. The first step to analyzing this algorithm is to use a stability lemma, essentially proved in Theorem 1.1 of [9], which bounds the regret as follows:

$$\text{Regret}_T \leq \sum_{t=1}^{T} f_t(S_t) - f_t(S_{t+1}) + R(S^*) - R(S_1).$$

Here, $S^* = \arg\min_{S \subseteq [n]} \sum_{t=1}^{T} f_t(S)$.

To bound the expected regret, by linearity of expectation, it suffices to bound $\mathbf{E}[f(S_t) - f(S_{t+1})]$, where for the purpose of analysis, we assume that we re-randomize in every round (i.e. choose a fresh random function $R : 2^{[n]} \to \mathbb{R}$). Naturally, the expectation $\mathbf{E}[f(S_t) - f(S_{t+1})]$ is the same regardless of when $R$ is chosen.

To bound this, we need the following lemma:

**Lemma 5.**
$$\mathbf{Pr}[S_t \neq S_{t+1}] \leq 2n\eta.$$

*Proof.* First, we note the following simple union bound:

$$\mathbf{Pr}[S_t \neq S_{t+1}] \leq \sum_{i \in [n]} \mathbf{Pr}[i \in S_t \text{ and } i \notin S_{t+1}] + \mathbf{Pr}[i \notin S_t \text{ and } i \in S_{t+1}]. \qquad (1)$$

Now, fix any $i$, and we aim to bound $\mathbf{Pr}[i \in S_t \text{ and } i \notin S_{t+1}]$. For this, we condition on the randomness in choosing $r_j$ for all $j \neq i$. Define $R' : 2^{[n]} \to \mathbb{R}$ as $R'(S) = \sum_{j \in S, j \neq i} r_j$, and $\Phi'_t : 2^{[n]} \to \mathbb{R}$ as $\Phi'_t(S) = \sum_{\tau=1}^{t-1} f_\tau(S) + R'(S)$. Note that if $i \notin S$, then $R'(S) = R(S)$ and $\Phi'_t(S) = \Phi_t(S)$. Let

$$A = \arg\min_{S \subseteq [n]: i \in S} \Phi'(S) \quad \text{and} \quad B = \arg\min_{S \subseteq [n]: i \notin S} \Phi'(S).$$

Now, we note that the event $i \in S_t$ happens only if $\Phi'_t(A) + r_i < \Phi'_t(B)$, and $S_t = A$. But if $\Phi'_t(A) + r_i < \Phi'_t(B) - 2$, then we must have $i \in S_{t+1}$, since for any $C$ such that $i \notin C$,

$$\Phi_{t+1}(A) = \Phi'_t(A) + r_i + f_t(A) < \Phi'_t(B) - 1 < \Phi'_t(C) + f_t(C) = \Phi_t(C).$$

The inequalities above use the fact that $f_t(S) \in [-1, 1]$ for all $S \subseteq [n]$. Thus, if $v := \Phi'_t(B) - \Phi'_t(A)$, we have

$$\mathbf{Pr}[i \in S_t \text{ and } i \notin S_{t+1} \mid r_j, j \neq i] \leq \mathbf{Pr}[r_i \in [v - 2, v] \mid r_j, j \neq i] \leq \eta,$$

since $r_i$ is chosen uniformly from $[-1/\eta, 1/\eta]$. We can now remove the conditioning on $r_j$ for $j \neq i$, and conclude that

$$\mathbf{Pr}[i \in S_t \text{ and } i \notin S_{t+1}] \leq \eta.$$

Similarly, we can bound $\mathbf{Pr}[i \notin S_t \text{ and } i \in S_{t+1}] \leq \eta$. Finally, the union bound (1) over all choices of $i$ yields the required bound on $\mathbf{Pr}[S_t \neq S_{t+1}]$. $\qquad \square$

Continuing the proof, we have

$$
\begin{aligned}
\mathbf{E}[f(S_t) - f(S_{t+1})] &= \mathbf{E}[f(S_t) - f(S_{t+1}) \mid S_t \neq S_{t+1}] \cdot \mathbf{Pr}[S_t \neq S_{t+1}] \\
&\leq 0 + 2 \cdot \mathbf{Pr}[S_t \neq S_{t+1}] \\
&\leq 4n\eta.
\end{aligned}
$$

The last inequality follows from Lemma 5. Now, we have $R(S^*) - R(S_1) \leq 2n/\eta$, and so

$$
\begin{aligned}
\mathbf{E}[\mathrm{Regret}_T] &\leq \sum_{t=1}^{T} \mathbf{E}[f(S_t) - f(S_{t+1})] + \mathbf{E}[R(S^*) - R(S_1)] \\
&\leq 4n\eta T + 2n/\eta \\
&\leq 6n\sqrt{T},
\end{aligned}
$$

since $\eta = 1/\sqrt{T}$. $\qquad\qquad\qquad\qquad\qquad\qquad\qquad\qquad\qquad\qquad\qquad\qquad\quad\square$

## 3.2 An Analytical Algorithm

In this section, we give a different algorithm based on the Online Gradient Descent method of Zinkevich [15]. We apply this technique to the Lovász extension of the cost function coupled with a simple randomized construction of the subgradient, as given in definition 2. This algorithm requires the concept of a *Euclidean projection* of a point in $\mathbb{R}^n$ on to the set $\mathcal{K}$, which is a function $\Pi_{\mathcal{K}} : \mathbb{R}^n \to \mathcal{K}$ defined by

$$
\Pi_{\mathcal{K}}(y) := \arg\min_{x \in \mathcal{K}} \|x - y\|.
$$

Since $\mathcal{K} = [0,1]^n$, it is easy to implement this projection: indeed, for a point $y \in \mathbb{R}^n$, the projection $x = \Pi_{\mathcal{K}}(y)$ is defined by

$$
x_i = \begin{cases} y_i & \text{if } y_i \in [0,1] \\ 0 & \text{if } y_i < 0 \\ 1 & \text{if } y_i > 1 \end{cases}
$$

---

**Algorithm 2** Submodular Subgradient Descent

1: Input: parameter $\eta > 0$. Let $x_1 \in \mathcal{K}$ be an arbitrary initial point.
2: **for** $t = 1$ to $T$ **do**
3:      Choose a threshold $\tau \in [0,1]$ uniformly at random, and use the set $S_t = \{i : x_t(i) > \tau\}$ and obtain cost $f_t(S_t)$.
4:      Find a maximal chain associated with $x_t$, $\emptyset = B_0 \subset B_1 \subset B_2 \subset \cdots B_n = [n]$, and use $f_t(B_0), f_t(B_1), \ldots, f_t(B_n)$ to compute a subgradient $g_t$ of $\hat{f}_t$ at $x_t$ as in part 2 of Proposition 3.
5:      Update: set $x_{t+1} = \Pi_{\mathcal{K}}(x_t - \eta g_t)$.
6: **end for**

---

In the analysis of the algorithm, we need the following regret bound. It is a simple extension of Zinkevich's analysis of Online Gradient Descent to vector-valued random variables whose expectation is the subgradient of the cost function (the generality to random variables is not required for this section, but it will be useful in the next section):

**Lemma 6.** *Let* $\hat{f}_1, \hat{f}_2, \ldots, \hat{f}_T : \mathcal{K} \to [-1, 1]$ *be a sequence of convex cost functions over the cube* $\mathcal{K}$. *Let* $x_1, x_2, \ldots, x_T \in \mathcal{K}$ *be defined by* $x_1 = 0$ *and* $x_{t+1} = \Pi_{\mathcal{K}}(x_t - \eta \hat{g}_t)$, *where* $\hat{g}_1, \hat{g}_2, \ldots, \hat{g}_T$ *are vector-valued random variables such that* $\mathbf{E}[\hat{g}_t | x_t] = g_t$, *where* $g_t$ *is a subgradient of* $\hat{f}_t$ *at* $x_t$. *Then the expected regret of playing* $x_1, x_2, \ldots, x_T$ *is bounded by*

$$
\sum_{t=1}^{T} \mathbf{E}[\hat{f}_t(x_t)] - \min_{x \in \mathcal{K}} \sum_{t=1}^{T} \hat{f}_T(x) \leq \frac{n}{2\eta} + 2\eta n \sum_t \mathbf{E}[\|\hat{g}_t\|^2].
$$

Since this Lemma follows rather easily from [15], we omit the proof in this extended abstract.

We can now prove the following regret bound:

**Theorem 7.** *Algorithm 2, run with parameter $\eta = 1/\sqrt{T}$, achieves the following regret bound:*

$$\mathbf{E}[Regret_T] \leq 3n\sqrt{T}.$$

*Furthermore, with probability at least $1 - \varepsilon$, $Regret_T \leq (3n + \sqrt{2\log(1/\varepsilon)})\sqrt{T}$.*

*Proof.* Note that be Definition 2, we have that $\mathbf{E}[f_t(S_t)] = \hat{f}_t(x_t)$. Since the algorithm runs Online Gradient Descent (from Lemma 6) with $\hat{g}_t = g_t$ (i.e. no randomness), we get the following bound on the regret. Here, we use the bound $\|\hat{g}_t\|^2 = \|g_t\|^2 \leq 4n$.

$$\mathbf{E}[Regret_T] \;=\; \sum_{t=1}^{T} \mathbf{E}[f_t(S_t)] - \min_{S \subseteq [n]} \sum_{t=1}^{T} f(S) \;\leq\; \sum_{t=1}^{T} \hat{f}_t(x_t) - \min_{x \in \mathcal{K}} \sum_{t=1}^{T} \hat{f}_T(x) \;\leq\; \frac{n}{2\eta} + 2\eta nT.$$

Since $\eta = 1/\sqrt{T}$, we get the required regret bound. Furthermore, by a simple Hoeffding bound, we also get that with probability at least $1 - \varepsilon$,

$$\sum_{t=1}^{T} f_t(S_t) \;\leq\; \sum_{t=1}^{T} \mathbf{E}[f_t(S_t)] + \sqrt{2T\log(1/\varepsilon)},$$

which implies the high probability regret bound. $\qquad\square$

# 4 The Bandit Setting

We now present an algorithm for the Bandit Online Submodular Minimization problem. The algorithm is based on the Online Gradient Descent algorithm of Zinkevich [15]. The main idea is use just one sample for both exploration (to construct an unbiased estimator for the subgradient) and exploitation (to construct an unbiased estimator for the point chosen by the Online Gradient Descent algorithm).

---

**Algorithm 3** Bandit Submodular Subgradient Descent

---

1: Input: parameters $\eta, \delta > 0$. Let $x_1 \in \mathcal{K}$ be arbitrary.
2: **for** $t = 1$ to $T$ **do**
3:     Find a maximal chain associated with $x_t$, $\emptyset = B_0 \subset B_1 \subset B_2 \subset \cdots B_n = [n]$, and let $\pi$ be the associated permutation as in part 2 of Proposition 3. Then $x_t$ can be written as $x_t = \sum_{i=0}^{n} \mu_i \chi_{B_i}$, where $\mu_i = 0$ for the extra sets $B_i$ that were added to complete the maximal chain for $x_t$.
4:     Choose the set $S_t$ as follows:

$$S_t = B_i \quad \text{with probability} \quad \rho_i = (1 - \delta)\mu_i + \frac{\delta}{n+1}.$$

    Use the set $S_t$ and obtain cost $f_t(S_t)$.
5:     If $S_t = B_0$, then set $\hat{g}_t = -\frac{1}{\rho_0} f_t(S_t) e_{\pi(1)}$, and if $S_t = B_n$ then set $\hat{g}_t = \frac{1}{\rho_n} f_t(S_t) e_{\pi(n)}$. Otherwise, $S_t = B_i$ for some $i \in [2, n-1]$. Choose $\varepsilon_t \in \{+1, -1\}$ uniformly at random, and set:

$$\hat{g}_t = \begin{cases} \frac{2}{\rho_i} f_t(S_t) e_{\pi(i)} & \text{if } \varepsilon_t = 1 \\ -\frac{2}{\rho_i} f_t(S_t) e_{\pi(i+1)} & \text{if } \varepsilon_t = -1 \end{cases}$$

6:     Update: set $x_{t+1} = \Pi_K(x_t - \eta \hat{g}_t)$.
7: **end for**

---

Before launching into the analysis, we define some convenient notation first. For a random variable $X_t$ defined in round $t$ of the algorithm, define $\mathbf{E}_t[X_t]$ (resp. $\text{VAR}_t[X_t]$) to be the expectation (resp. variance) of $X_t$ conditioned on all the randomness chosen by the algorithm until round $t$.

A first observation is that on the expectation, the regret of the algorithm above is almost the same as if it had played $x_t$ all along and the loss functions were replaced by the Lovász extensions of the actual loss functions.

**Lemma 8.** *For all t, we have* $\mathbf{E}[f(S_t)] \leq \mathbf{E}[\hat{f}_t(x_t)] + 2\delta$.

*Proof.* From Definition 2 we have that $\hat{f}(x_t) = \sum_i \mu_i f(B_i)$. On the other hand, $\mathbf{E}_t[f(S_t)] = \sum_i \rho_i f(B_i)$, and hence:

$$\mathbf{E}_t[f(S_t)] - \hat{f}_t(x_t) = \sum_{i=0}^n (\rho_i - \mu_i) f(B_i) \leq \delta \sum_{i=0}^n \left[\frac{1}{n+1} + \mu_i\right] |f(B_i)| \leq 2\delta.$$

The lemma now follows by taking the expectation of both sides of this inequality with respect to the randomness chosen in the first $t-1$ rounds. $\square$

Next, by Proposition 3, the subgradient of the Lovász extension of $f_t$ at point $x_t$ corresponding to the maximal chain $B_0 \subset B_1 \subset \cdots \subset B_n$ is given by $g_t(i) = f(B_{\pi(i)}) - f(B_{\pi(i)-1})$. Using this fact, it is easy to check that the random vector $\hat{g}_t$ is constructed in such a way that $\mathbf{E}[\hat{g}_t|x_t] = g_t$. Furthermore, we can bound the norm of this estimator as follows:

$$\mathbf{E}_t[\|\hat{g}_t\|^2] \leq \sum_{i=0}^n \frac{4}{\rho_i^2} f_t(B_i)^2 \cdot \rho_i \leq \frac{4(n+1)^2}{\delta} \leq \frac{16n^2}{\delta}. \tag{2}$$

We can now remove the conditioning, and conclude that $\mathbf{E}[\|\hat{g}_t\|^2] \leq \frac{16n^2}{\delta}$.

**Theorem 9.** *Algorithm 3, run with parameters* $\delta = \frac{n}{T^{1/3}}$, $\eta = \frac{1}{T^{2/3}}$, *achieves the following regret bound:*

$$\mathbf{E}[Regret_T] \leq 12nT^{2/3}.$$

*Proof.* We bound the expected regret as follows:

$$\sum_{t=1}^T \mathbf{E}[f_t(S_t)] - \min_{S \subseteq [n]} \sum_{t=1}^T f_t(S) \leq 2\delta T + \sum_{t=1}^T \mathbf{E}[\hat{f}_t(x_t)] - \min_{x \in \mathcal{K}} \sum_{t=1}^T \hat{f}_t(x) \quad \text{(By Lemma 8)}$$

$$\leq 2\delta T + \frac{n}{2\eta} + \frac{\eta}{2} \sum_{t=1}^T \mathbf{E}[\|\hat{g}_t\|^2] \quad \text{(By Lemma 6)}$$

$$\leq 2\delta T + \frac{n}{2\eta} + \frac{8n^2\eta T}{\delta}. \quad \text{(By (2))}$$

The bound is now obtained for $\delta = \frac{n}{T^{1/3}}$, $\eta = \frac{1}{T^{2/3}}$. $\square$

## 4.1 High probability bounds on the regret

The theorem of the previous section gave a bound on the expected regret. However, a much stronger claim can be made that essentially the same regret bound holds with very high probability (exponential tail). In addition, the previous theorem (which only bounds expected regret) holds against an oblivious adversary, but not necessarily against a more powerful adaptive adversary. The following gives high probability bounds against an adaptive adversary.

**Theorem 10.** *With probability* $1 - 4\varepsilon$, *Algorithm 3, run with parameters* $\delta = \frac{n}{T^{1/3}}$, $\eta = \frac{1}{T^{2/3}}$, *achieves the following regret bound:*

$$Regret_T \leq O(nT^{2/3}\sqrt{\log(1/\varepsilon)}).$$

The proof of this theorem is deferred to the full version of this paper.

## 5 Conclusions and Open Questions

We have described efficient regret minimization algorithms for submodular cost functions, in both the bandit and full information settings. This parallels the work of Streeter and Golovin [14] who study two specific instances of online submodular *maximization* (for which the offline problem is NP-hard), and give (approximate) regret minimizing algorithms. An open question is whether it is possible to attain $O(\sqrt{T})$ regret bounds for online submodular minimization in the bandit setting.

## Footnotes

[1]Note that this is not the standard definition of the Lovász extension, but an equivalent characterization.

# References

[1] A. D. Flaxman, A. T. Kalai, and H. B. McMahan, *Online convex optimization in the bandit setting: gradient descent without a gradient*, SODA, 2005, pp. 385–394.

[2] Satoru Fujishige, *Submodular functions and optimization*, Elsevier, 2005.

[3] M. Grötschel, L. Lovász, and A. Schrijver, *Geometric Algorithms and Combinatorial Optimization*, Springer Verlag, 1988.

[4] Carlos Guestrin and Andreas Krause, *Beyond convexity - submodularity in machine learning.*, Tutorial given in the 25rd International Conference on Machine Learning (ICML), 2008.

[5] J. Hannan, *Approximation to bayes risk in repeated play*, In M. Dresher, A. W. Tucker, and P. Wolfe, editors, Contributions to the Theory of Games, volume III (1957), 97–139.

[6] Satoru Iwata, *A faster scaling algorithm for minimizing submodular functions*, SIAM J. Comput. **32** (2003), no. 4, 833–840.

[7] Satoru Iwata, Lisa Fleischer, and Satoru Fujishige, *A combinatorial strongly polynomial algorithm for minimizing submodular functions*, J. ACM **48** (2001), 761–777.

[8] Satoru Iwata and James B. Orlin, *A simple combinatorial algorithm for submodular function minimization*, SODA '09: Proceedings of the Nineteenth Annual ACM -SIAM Symposium on Discrete Algorithms (Philadelphia, PA, USA), Society for Industrial and Applied Mathematics, 2009, pp. 1230–1237.

[9] Adam Kalai and Santosh Vempala, *Efficient algorithms for online decision problems*, Journal of Computer and System Sciences **71(3)** (2005), 291–307.

[10] N. Littlestone and M. K. Warmuth, *The weighted majority algorithm*, Proceedings of the 30th Annual Symposium on the Foundations of Computer Science, 1989, pp. 256–261.

[11] S. T. McCormick, *Submodular function minimization.*, Chapter 7 in the Handbook on Discrete Optimization (G. Nemhauser K. Aardal and R. Weismantel, eds.), Elsevier, 2006, pp. 321–391.

[12] James B. Orlin, *A faster strongly polynomial time algorithm for submodular function minimization*, Math. Program. **118** (2009), no. 2, 237–251.

[13] Alexander Schrijver, *A combinatorial algorithm minimizing submodular functions in strongly polynomial time*, 1999.

[14] Matthew J. Streeter and Daniel Golovin, *An online algorithm for maximizing submodular functions*, NIPS, 2008, pp. 1577–1584.

[15] Martin Zinkevich, *Online convex programming and generalized infinitesimal gradient ascent.*, Proceedings of the Twentieth International Conference (ICML), 2003, pp. 928–936.

